# Correlational Strength and Computational Algebra
# of Synaptic Connections Between Neurons

Eberhard E. Fetz
Department of Physiology & Biophysics,
University of Washington, Seattle, WA 98195

## ABSTRACT

Intracellular recordings in spinal cord motoneurons and cerebral cortex neurons have provided new evidence on the correlational strength of monosynaptic connections, and the relation between the shapes of postsynaptic potentials and the associated increased firing probability. In these cells, excitatory postsynaptic potentials (EPSPs) produce cross-correlogram peaks which resemble in large part the derivative of the EPSP. Additional synaptic noise broadens the peak, but the peak area -- i.e., the number of above-chance firings triggered per EPSP -- remains proportional to the EPSP amplitude. A typical EPSP of 100 μv triggers about .01 firings per EPSP. The consequences of these data for information processing by polysynaptic connections is discussed. The effects of sequential polysynaptic links can be calculated by convolving the effects of the underlying monosynaptic connections. The net effect of parallel pathways is the sum of the individual contributions.

## INTRODUCTION

Interactions between neurons are determined by the strength and distribution of their synaptic connections. The strength of synaptic interactions has been measured directly in the central nervous system by two techniques. Intracellular recording reveals the magnitude and time course of postsynaptic potentials (PSPs) produced by synaptic connections, and cross-correlation of extracellular spike trains measures the effect of the PSP's on the firing probability of the connected cells. The relation between the shape of excitatory postsynaptic potentials (EPSPs) and the shape of the cross-correlogram peak they produce has been empirically investigated in cat motoneurons [2,4,5] and in neocortical cells [10].

### RELATION BETWEEN EPSP'S AND CORRELOGRAM PEAKS

Synaptic interactions have been studied most thoroughly in spinal cord motoneurons. Figure 1 illustrates the membrane potential of a rhythmically firing motoneuron, and the effect of EPSPs on its firing. An EPSP occurring sufficiently close to threshold (Θ) will cause the motoneuron to fire and will advance an action potential to its rising edge (top). Mathematical analysis of this threshold-crossing process predicts that an EPSP with shape e(t) will produce a firing probability f(t), which resembles

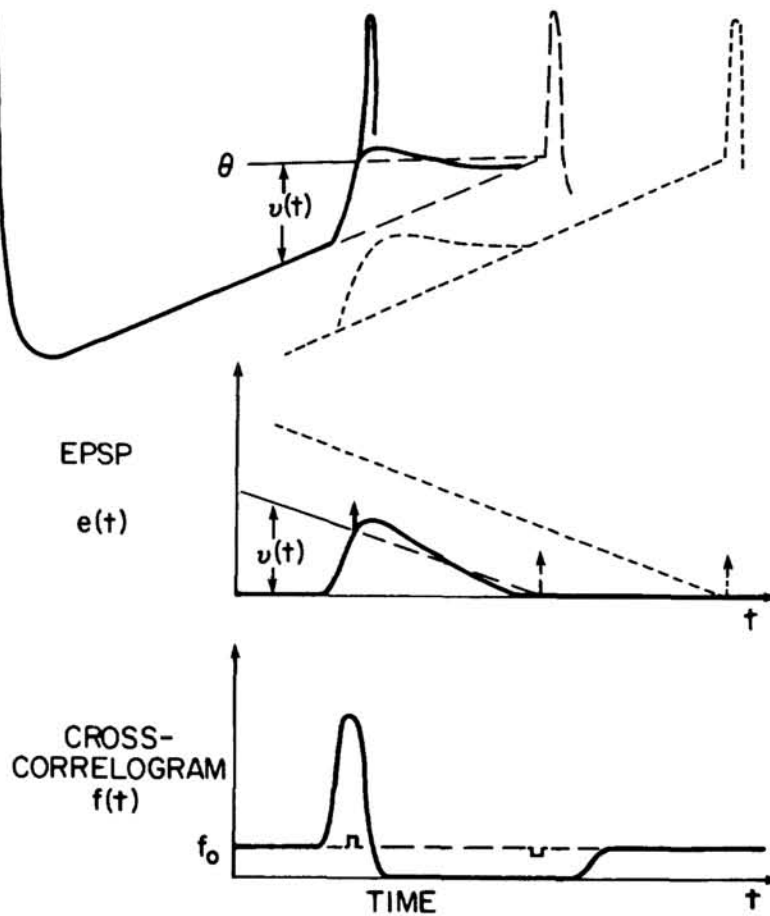

EPSP

e(t)

CROSS-
CORRELOGRAM
f(t)

TIME

Fig. 1. The relation between EPSP's and motoneuron firing. Top: membrane trajectory of rhythmically firing motoneuron, showing EPSP crossing threshold ($\Theta$) and shortening the normal interspike interval by advancing a spike. V(t) is difference between membrane potential and threshold. Middle: same threshold-crossing process aligned with EPSP, with v(t) plotted as falling trajectory. Intercept (at upward arrow) indicates time of the advanced action potential. Bottom: Cross-correlation histogram predicted by threshold crossings. The peak in the firing rate f(t) above baseline ($f_0$) is produced by spikes advanced from baseline, as indicated by the changed counts for the illustrated trajectory. Consequently, the area in the peak equals the area of the subsequent trough.

the derivative of the EPSP [4,8]. Specifically, for smooth membrane potential trajectories approaching threshold (the case of no additional synaptic noise):

$$f(t) = f_0 + (f_0/\dot{v})\, de/dt \qquad (1)$$

where $f_0$ is the baseline firing rate of the motoneuron and $\dot{v}$ is the rate of closure between motoneuron membrane potential and threshold. This relation can be derived analytically by tranforming the process to a coordinate system aligned with the EPSP (Fig. 1, middle) and calculating the relative timing of spikes advanced by intercepts of the threshold trajectories with the EPSP [4]. The above relation (1) is also valid for the correlogram trough during the falling phase of the EPSP, as long as $de/dt > -\dot{v}$; if the EPSP falls more rapidly than $-\dot{v}$, the trough is limited at zero firing rate (as illustrated for the correlogram at bottom). The fact that the shape of the correlogram peak above baseline matches the EPSP derivative has been empirically confirmed for large EPSPs in cat motoneurons [4]. This relation implies that the height of the correlogram peak above baseline is proportional to the EPSP rate of rise. The integral of this relationship predicts that the area between the correlogram peak and baseline is proportional to the EPSP amplitude. This linear relation further implies that the effects of simultaneously arriving EPSPs will add linearly.

The presence of additional background synaptic "noise", which is normally produced by randomly occurring synaptic inputs, tends to make the correlogram peak broader than the duration of the EPSP risetime. This broadening is produced by membrane potential fluctuations which cause additional threshold crossings during the decay of the EPSP by trajectories that would have missed the EPSP (e.g., the dashed trajectory in Fig. 1, middle). On the basis of indirect empirical comparisons it has been proposed [6,7] that the broader correlogram peaks can be described by the sum of two linear functions of e(t):

$$f(t) = f_0 + a\, e(t) + b\, de/dt \qquad (2)$$

This relation provides a reasonable match when the coefficients (a and b) can be optimized for each case [5,7], but direct empirical comparisons [2,4] indicate that the difference between the correlogram peak and the derivative is typically briefer than the EPSP.

The effect of synaptic noise on the transform between EPSP and correlogram peak has not yet been analytically derived (except for the case of Gaussian noise[1]). However the threshold-crossing process has been simulated by a computer model which adds synaptic noise to the trajectories intercepting the EPSP [1]. The correlograms generated by the simulation match the correlograms measured empirically for small EPSP's in motoneurons [2], confirming the validity of the model.

Although synaptic noise distributes the triggered firings over a wider peak, the area of the correlogram peak, i.e., the number of motoneuron firings produced by an EPSP, is essentially preserved and remains proportional to EPSP amplitude for moderate noise levels. For unitary EPSP's (produced by

a single afferent fiber) in cat motoneurons, the number of firings triggered per EPSP ($N_p$) was linearly related to the amplitude (h) of the EPSP [2]:

$$N_p = (0.1/mv) \cdot h \ (mv) + .003 \qquad (3)$$

The fact that the number of triggered spikes increases in proportion to EPSP amplitude has also been confirmed for neocortical neurons [10]; for cells recorded in sensorimotor cortex slices (probably pyramidal cells) the coefficient of h was very similar: 0.07/mv. This means that a typical unitary EPSP with amplitude of 100 μv, raises the probability that the postsynaptic cell fires by less than .01. Moreover, this increase occurs during a specific time interval corresponding to the rise time of the EPSP -- on the order of 1 - 2 msec. The net increase in firing rate of the postsynaptic cell is calculated by the proportional decrease in interspike intervals produced by the triggered spikes [4]. (While the above values are typical, unitary EPSP's range in size from several hundred μv down to undetectable levels of several μv., and have risetimes of .2 - 4 msec.)

Inhibitory connections between cells, mediated by inhibitory postsynaptic potentials (IPSPs), produce a trough in the cross-correlogram. This reduction of firing probability below baseline is followed by a subsequent broad, shallow peak, representing the spikes that have been delayed during the IPSP. Although the effects of inhibitory connections remain to be analyzed more quantitatively, preliminary results indicate that small IPSP's in synaptic noise produce decreases in firing probability that are similar to the increases produced by EPSP's [4,5].

## DISYNAPTIC LINKS

The effects of polysynaptic links between neurons can be understood as combinations of the underlying monosynaptic connections. A monosynaptic connection from cell A to cell B would produce a first-order cross-correlation peak $P_1(B|A,t)$, representing the conditional probability that neuron B fires above chance at time t, given a spike in cell A at time t = 0. As noted above, the shape of this first-order correlogram peak is largely proportional to the EPSP derivative (for cells whose interspike interval exceeds the duration of the EPSP). The latency of the peak is the conduction time from A to B (Fig. 2 top left).

In contrast, several types of disynaptic linkages between A and B, mediated by a third neuron C, will produce a second-order correlation peak between A and B. A disynaptic link may be produced by two serial monosynaptic connections, from A to C and from C to B (Fig. 2, bottom left), or by a common synaptic input from C ending on both A and B (Fig. 2, bottom right). In both cases, the second-order correlation between A and B produced by the disynaptic link would be the convolution of the two first-order correlations between the monosynaptically connected cells:

$$P_2(B|A) = P_1(B|C) \otimes P_1(C|A) \qquad (4)$$

As indicated by the diagram, the cross-correlogram peak $P_2(B|A,t)$ would be smaller and more dispersed than the peaks of the underlying first-order correlation peaks. For serial connections the peak would appear to the right of the origin, at a latency that is the sum of the two monosynaptic latencies. The peak produced by a common input typically straddles the origin, since its timing reflects the difference between the underlying latencies.

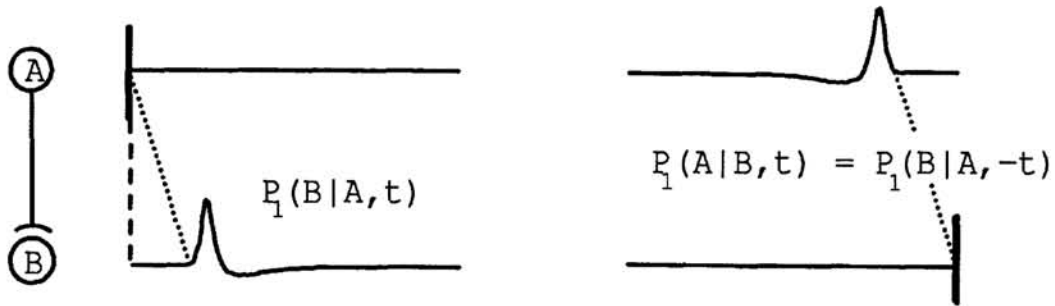

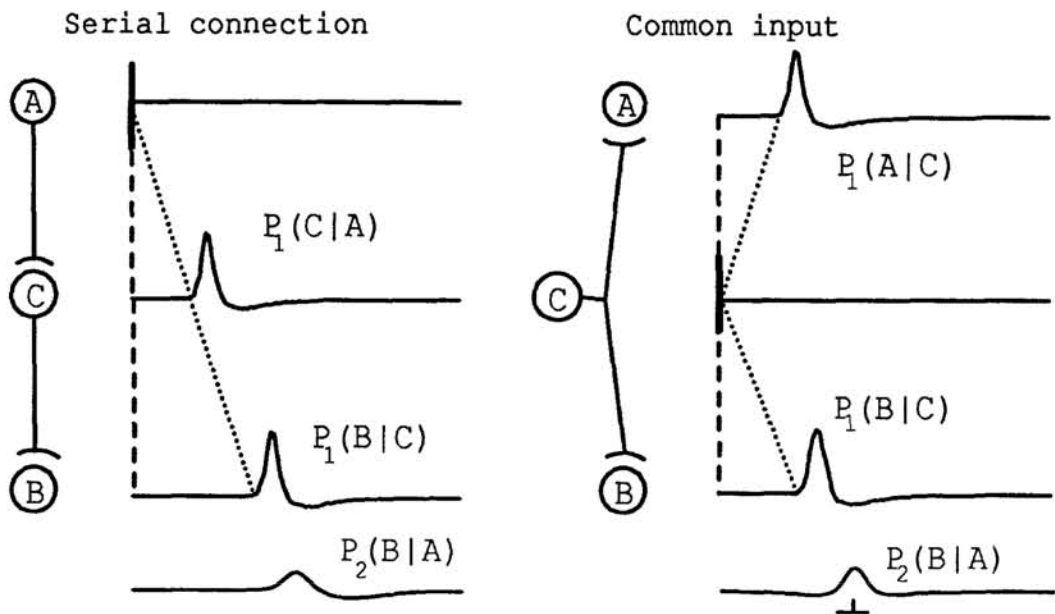

Fig. 2. Correlational effects of monosynaptic and disynaptic links between two neurons. Top: monosynaptic excitatory link from A to B produces an increase in firing probability of B after A (left). As with all correlograms this is the time-inverted probability of increased firing in A relative to B (right). Bottom: Two common disynaptic links between A and B are a serial connection via C (left) and a common input from C. In both cases the effect of the disynaptic link is the convolution of the underlying monosynaptic links.

This relation means that the probability that a spike in cell A will produce a correlated spike in cell B would be the product of the two probabilities for the intervening monosynaptic connections. Given a typical $N_p$ of .01/EPSP, this would reduce the effectiveness of a given disynaptic linkage by two orders of magnitude relative to a monosynaptic connection. However, the net strength of *all* the disynaptic linkages between two given cells is proportional to the number of mediating interneurons {C}, since the effects of parallel pathways add. Thus, the net potency of all the disynaptic linkages between two cells could approach that of a monosynaptic linkage if the number of mediating interneurons were sufficiently large. It should also be noted that some interneurons may fire more than once per EPSP and have a higher probability of being triggered to fire than motoneurons [11].

For completeness, two other possible disynaptic links between A and B involving a third cell C may be considered. One is a serial connection from B to C to A, which is the reverse of the serial connection from A to B. This would produce a $P_2(B|A)$ with peak to the left of the origin. The fourth circuit involves convergent connections from both A and B to C; this is the only combination that would not produce any causal link between A and B.

The effects of still higher-order polysynaptic linkages can be computed similarly, by convolving the effects produced by the sequential connections. For example, trisynaptic linkages between four neurons are equivalent to combinations of disynaptic and monosynaptic connections.

The cross-correlograms between two cells have a certain symmetry, depending on which is the reference cell. The cross-correlation histogram of cell B referenced to A is identical to the time-inverted correlogram of A referenced to B. This is illustrated for the monosynaptic connection in Fig.2, top right, but is true for all correlograms. This symmetry represents the fact that the above-chance probability of B firing after A is the same as the probability of A firing before B:

$$P(B|A, t) = P(A|B, -t) \qquad (5)$$

As a consequence, polysynaptic correlational links can be computed as the same convolution integral (Eq. 4), independent of the direction of impulse propagation.

## PARALLEL PATHS AND FEEDBACK LOOPS

In addition to the simple combinations of pair-wise connections between neurons illustrated above, additional connections between the same cells may form circuits with various kinds of loops. Recurrent connections can produce feedback loops, whose correlational effects are also calculated by convolving effects of the underlying synaptic links. Parallel feed-forward paths can form multiple pathways between the same cells. These produce correlational effects that are the sum of the effects of the individual underlying connections.

The simplest feedback loop is formed by reciprocal connections between a pair of cells. The effects of excitatory feedback can be computed by

successive convolutions of the underlying monosynaptic connections (Fig. 3 top). Note that such a positive feedback loop would be capable of sustaining activity only if the connections were sufficiently potent to ensure postsynaptic firing. Since the probabilities of triggered firings at a single synapse are considerably less than one, reverberating activity can be sustained only if the number of interacting cells is correspondingly increased. Thus, if the probability for a single link is on the order of .01, reverberating activity can be sustained if A and B are similarly interconnected with at least a hundred cells in parallel.

Connections between three neurons may produce various kinds of loops. *Feedforward parallel pathways* are formed when cell A is monosynaptically connected to B and in addition has a serial disynaptic connection through C, as illustrated in Fig. 3 (bottom left); the correlational effects of the two linkages from A to B would sum linearly, as shown for excitatory connections. Again, the effect of a larger set of cells {C} would be additive. *Feedback loops* could be formed with three cells by recurrent connections between any pair; the correlational consequences of the loop again are the convolution of the underlying links. Three cells can form another type loop if both A and B are monosynaptically connected, and simultaneously influenced by a common interneuron C (Fig. 3 bottom right). In this case the expected correlogram between A and B would be the sum of the individual components -- a common input peak around the origin plus a delayed peak produced by the serial connection.

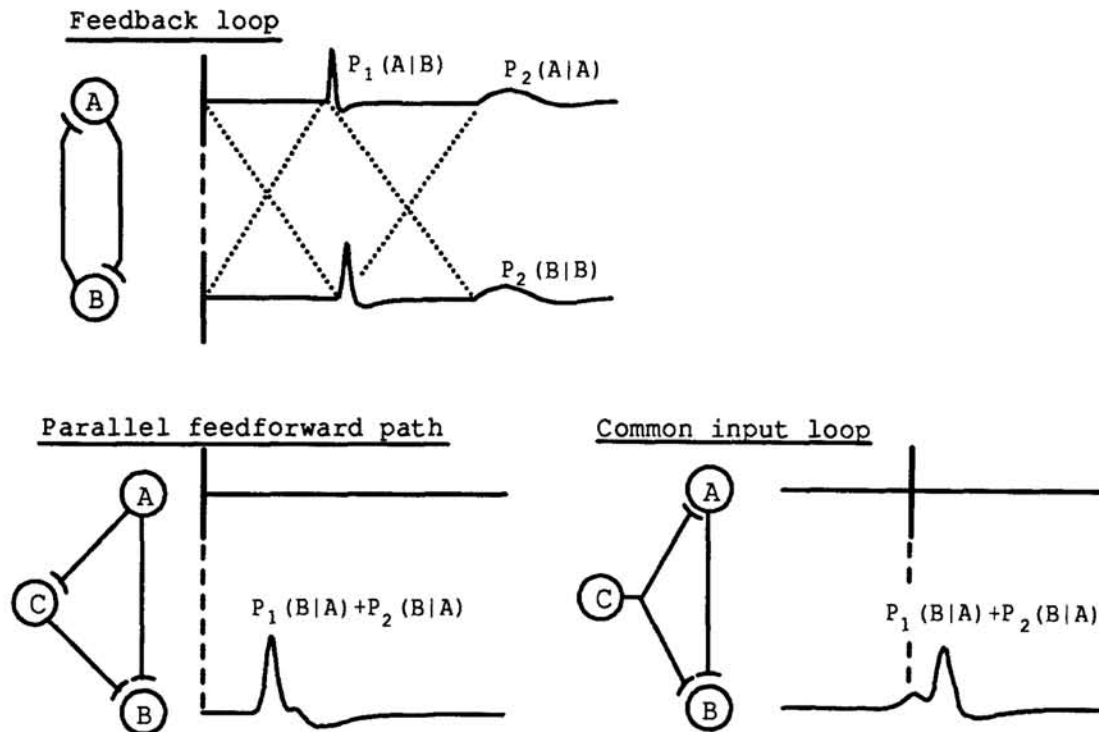

Fig. 3. Correlational effects of parallel connections between two neurons. Top: feedback loop between two neurons A and B produces higher-order effects equivalent to convolution of monosynaptic effects. Bottom: Loops formed by parallel feedforward paths (left) and by a common input concurrent with a monosynaptic link (right) produce additive effects.

## CONCLUSIONS

Thus, a simple computational algebra can be used to derive the correlational effects of a given network structure. Effects of sequential connections can be computed by convolution and effects of parallel paths by summation. The inverse problem, of deducing the circuitry from the correlational data is more difficult, since similar correlogram features may be produced by different circuits [9].

The fact that monosynaptic links produce small correlational effects on the order of .01 represents a significant constraint in the mechanisms of information processing in real neural nets. For example, secure propagation of activity through serial polysynaptic linkages requires that the small probability of triggered firing via a given link is compensated by a proportional increase in the number of parallel links. Thus, reliable serial conduction would require hundreds of neurons at each level, with appropriate divergent and convergent connections. It should also be noted that the effect of interneurons can be modulated by changing their activity. The intervening cells need to be active to mediate the correlational effects. As indicated by eq. 1, the size of the correlogram peak is proportional to the firing rate ($f_o$) of the postsynaptic cell. This allows dynamic modulation of polysynaptic linkages. The greater the number of links, the more susceptible they are to modulation.

Acknowledgements: The author thanks Mr. Garrett Kenyon for stimulating discussions and the cited colleagues for collaborative efforts. This work was supported in part by NIH grants NS 12542 and RR00166.

## REFERENCES

1. Bishop, B., Reyes, A.D., and Fetz E.E., Soc. for Neurosci Abst. 11:157 (1985).
2. Cope, T.C., Fetz, E.E., and Matsumura, M., J. Physiol. 390:161-18 (1987).
3. Fetz, E.E. and Cheney, P.D., J. Neurophysiol. 44:751-772 (1980).
4. Fetz, E.E. and Gustafsson, B., J. Physiol. 341:387-410 (1983).
5. Gustafsson, B., and McCrea, D., J. Physiol. 347:431-451 (1984).
6. Kirkwood, P.A., J. Neurosci. Meth. 1:107-132 (1979).
7. Kirkwood, P.A., and Sears, T. J. Physiol. 275:103-134 (1978).
8. Knox, C.K., Biophys. J. 14: 567-582 (1974).
9. Moore, G.P., Segundo, J.P., Perkel, D.H. and Levitan, H., Biophys. J. 10:876-900 (1970).
10. Reyes, A.D., Fetz E.E. and Schwindt, P.C., Soc. for Neurosci Abst. 13:157 (1987).
11. Surmeier, D.J. and Weinberg, R.J., Brain Res. 331:180-184 (1985).